# Adaptive Synchronization of Neural and Physical Oscillators

**Kenji Doya**
University of California, San Diego
La Jolla, CA 92093-0322, USA

**Shuji Yoshizawa**
University of Tokyo
Bunkyo-ku, Tokyo 113, Japan

## Abstract

Animal locomotion patterns are controlled by recurrent neural networks called central pattern generators (CPGs). Although a CPG can oscillate autonomously, its rhythm and phase must be well coordinated with the state of the physical system using sensory inputs. In this paper we propose a learning algorithm for synchronizing neural and physical oscillators with specific phase relationships. Sensory input connections are modified by the correlation between cellular activities and input signals. Simulations show that the learning rule can be used for setting sensory feedback connections to a CPG as well as coupling connections between CPGs.

## 1 CENTRAL AND SENSORY MECHANISMS IN LOCOMOTION CONTROL

Patterns of animal locomotion, such as walking, swimming, and flying, are generated by recurrent neural networks that are located in segmental ganglia of invertebrates and spinal cords of vertebrates (Barnes and Gladden, 1985). These networks can produce basic rhythms of locomotion without sensory inputs and are called central pattern generators (CPGs). The physical systems of locomotion, such as legs, fins, and wings combined with physical environments, have their own oscillatory characteristics. Therefore, in order to realize efficient locomotion, the frequency and the phase of oscillation of a CPG must be well coordinated with the state of the physical system. For example, the bursting patterns of motoneurons that drive a leg muscle must be coordinated with the configuration of the leg, its contact with the ground, and the state of other legs.

The oscillation pattern of a CPG is largely affected by proprioceptive inputs. It has been shown in crayfish (Siller et al., 1986) and lamprey (Grillner et al, 1990) that the oscillation of a CPG is entrained by cyclic stimuli to stretch sensory neurons over a wide range of frequency. Both negative and positive feedback pathways are found in those systems. Elucidation of the function of the sensory inputs to CPGs requires computational studies of neural and physical dynamical systems. Algorithms for the learning of rhythmic patterns in recurrent neural networks have been derived by Doya and Yoshizawa (1989), Pearlmutter (1989), and Williams and Zipser (1989). In this paper we propose a learning algorithm for synchronizing a neural oscillator to rhythmic input signals with a specific phase relationship.

It is well known that a coupling between nonlinear oscillators can entrainment their frequencies. The relative phase between oscillators is determined by the parameters of coupling and the difference of their intrinsic frequencies. For example, either in-phase or anti-phase oscillation results from symmetric coupling between neural oscillators with similar intrinsic frequencies (Kawato and Suzuki, 1980). Efficient locomotion involves subtle phase relationships between physical variables and motor commands. Accordingly, our goal is to derive a learning algorithm that can finely tune the sensory input connections by which the relative phase between physical and neural oscillators is kept at a specific value required by the task.

## 2    LEARNING OF SYNCHRONIZATION

We will deal with the following continuous-time model of a CPG network.

$$\tau_i \frac{d}{dt} x_i(t) = -x_i(t) + \sum_{j=1}^{C} w_{ij} g_j(x_j(t)) + \sum_{k=1}^{S} v_{ik} y_k(t), \qquad (1)$$

where $x_i(t)$ and $g_i(x_i(t))$ $(i = 1, \ldots, C)$ represent the states and the outputs of CPG neurons and $y_k(t)$ $(k = 1, \ldots, S)$ represents sensory inputs. We assume that the connection weights $W = \{w_{ij}\}$ are already established so that the network oscillates without sensory inputs. The goal of learning is to find the input connection weights $V = \{v_{ij}\}$ that make the network state $\mathbf{x}(t) = (x_1(t), \ldots, x_C(t))^t$ entrained to the input signal $\mathbf{y}(t) = (y_1(t), \ldots, y_S(t))^t$ with a specific relative phase.

### 2.1    AN OBJECTIVE FUNCTION FOR PHASE-LOCKING

The standard way to derive a learning algorithm is to find out an objective function to be minimized. If we can approximate the waveforms of $x_i(t)$ and $y_k(t)$ by sine waves, a linear relationship

$$\mathbf{x}(t) = P\mathbf{y}(t)$$

specifies a phase-locked oscillation of $\mathbf{x}(t)$ and $\mathbf{y}(t)$. For example, if we have $y_1 = \sin \omega t$ and $y_2 = \cos \omega t$, then a matrix $P = \begin{pmatrix} 1 & 1 \\ 1 & \sqrt{3} \end{pmatrix}$ specifies $x_1 = \sqrt{2} \sin(\omega t + \pi/4)$ and $x_2 = 2 \sin(\omega t + \pi/3)$. Even when the waveforms are not sinusoidal, minimization of an objective function

$$E(t) = \frac{1}{2} \|\mathbf{x}(t) - P\mathbf{y}(t)\|^2 = \frac{1}{2} \sum_{i=1}^{C} \{x_i(t) - \sum_{k=1}^{S} p_{ik} y_k(t)\}^2 \qquad (2)$$

determines a specific relative phase between $\mathbf{x}(t)$ and $\mathbf{y}(t)$. Thus we call $P = \{p_{ik}\}$ a phase-lock matrix.

## 2.2   LEARNING PROCEDURE

Using the above objective function, we will derive a learning procedure for phase-locked oscillation of $\mathbf{x}(t)$ and $\mathbf{y}(t)$. First, an appropriate phase-lock matrix $P$ is identified while the relative phase between $\mathbf{x}(t)$ and $\mathbf{y}(t)$ changes gradually in time. Then, a feedback mechanism can be applied so that the network state $\mathbf{x}(t)$ is kept close to the target waveform $P\,\mathbf{y}(t)$.

Suppose we actually have an appropriate phase relationship between $\mathbf{x}(t)$ and $\mathbf{y}(t)$, then the phase-lock matrix $P$ can be obtained by gradient descent of $E(t)$ with respect to $p_{ik}$ as follows (Widrow and Stearns, 1985).

$$\frac{d}{dt}p_{ik} = -\eta\frac{\partial E(t)}{\partial p_{ik}} = \eta\left\{x_i(t) - \sum_{j=1}^{S} p_{ij}y_j(t)\right\}y_k(t). \tag{3}$$

If the coupling between $\mathbf{x}(t)$ and $\mathbf{y}(t)$ are weak enough, their relative phase changes in time unless their intrinsic frequencies are exactly equal and the systems are completely noiseless. By modulating the learning coefficient $\eta$ by some performance index of the total system, for example, the speed of locomotion, it is possible to obtain a matrix $P$ that satisfies the requirement of the task.

Once a phase-lock matrix is derived, we can control $\mathbf{x}(t)$ close to $P\,\mathbf{y}(t)$ using the gradient of $E(t)$ with respect to the network state

$$\frac{\partial E(t)}{\partial x_i(t)} = x_i(t) - \sum_{k=1}^{S} p_{ik}y_k(t).$$

The simplest feedback algorithm is to add this term to the CPG dynamics as follows.

$$\tau_i\frac{d}{dt}x_i(t) = -x_i(t) + \sum_{j=1}^{C} w_{ij}g_j(x_j(t)) - \alpha\{x_i(t) - \sum_{k=1}^{S} p_{ik}y_k(t)\}.$$

The feedback gain $\alpha\,(> 0)$ must be set small enough so that the feedback term does not destroy the intrinsic oscillation of the CPG. In that case, by neglecting the small additional decay term $\alpha x_i(t)$, we have

$$\tau_i\frac{d}{dt}x_i(t) = -x_i(t) + \sum_{j=1}^{C} w_{ij}g_j(x_j(t)) + \sum_{k=1}^{S} \alpha p_{ik}y_k(t), \tag{4}$$

which is equivalent to the equation (1) with input weights $v_{ik} = \alpha p_{ik}$.

## 3    DELAYED SYNCHRONIZATION

We tested the above learning scheme on a delayed synchronization task; to find coupling weights between neural oscillators so that they synchronize with a specific time delay. We used the following coupled CPG model.

$$\tau \frac{d}{dt} x_i^n(t) = -x_i^n(t) + \sum_{j=1}^{C} w_{ij}^n y_j^n(t) + \alpha \sum_{k=1}^{C} p_{ik}^n y_k^{3-n}(t), \qquad (5)$$

$$y_i^n(t) = g(x_i^n(t)), \qquad (i = 1, \ldots, C),$$

where superscripts denote the indices of two CPGs ($n = 1, 2$). The goal of learning was to synchronize the waveforms $y_1^1(t)$ and $y_1^2(t)$ with a time delay $\Delta T$. We used

$$z(t) = -|y_1^1(t - \Delta T) - y_1^2(t)|$$

as the performance index. The learning coefficient $\eta$ of equation (3) was modulated by the deviation of $z(t)$ from its running average $\bar{z}(t)$ using the following equations.

$$\eta(t) = \eta_0 \{z(t) - \bar{z}(t)\}, \qquad \tau_a \frac{d}{dt} \bar{z}(t) = -\bar{z}(t) + z(t). \qquad (6)$$

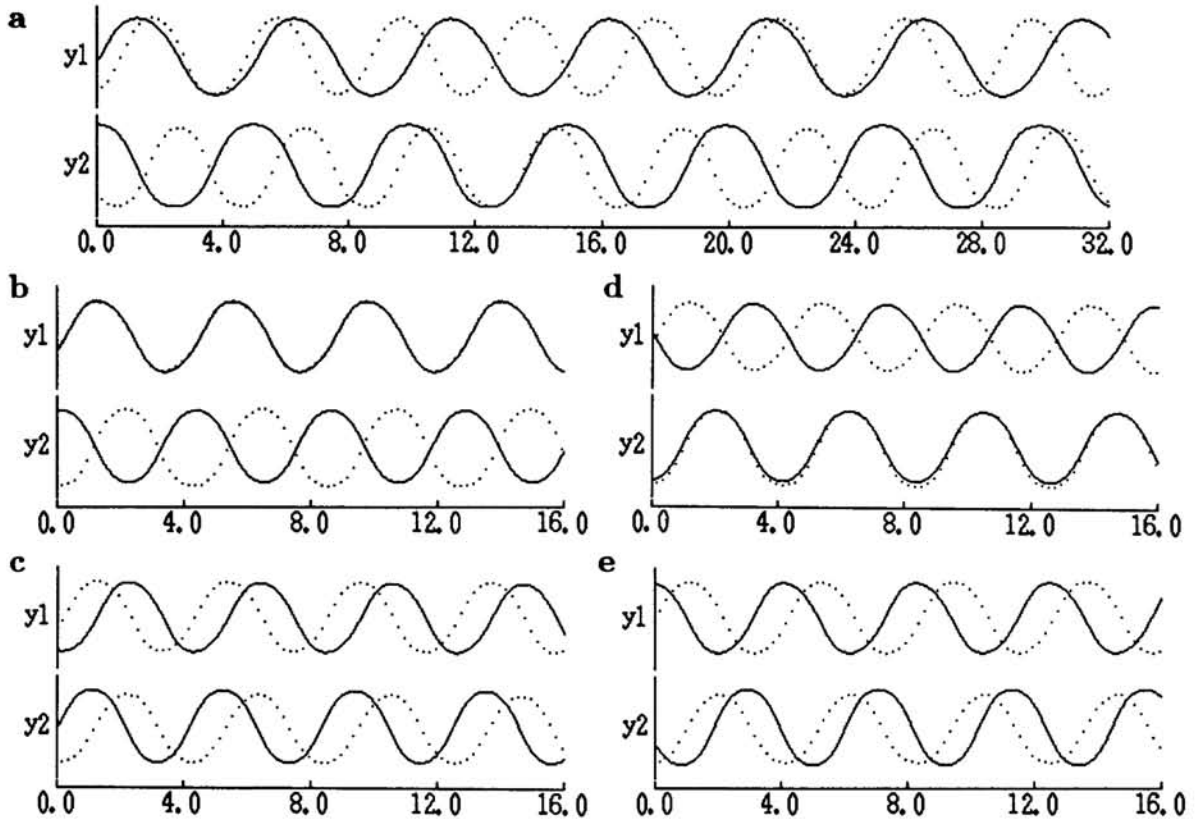

Figure 1: Learning of delayed synchronization of neural oscillators. The dotted and solid curves represent $y_i^1(t)$ and $y_i^2(t)$ respectively. a:without coupling. b:$\Delta T = 0.0$. c:$\Delta T = 1.0$. c:$\Delta T = 2.0$. d:$\Delta T = 3.0$.

First, two CPGs were trained independently to oscillate with sinusoidal waveforms of period $T_1 = 4.0$ and $T_2 = 5.0$ using continuous-time back-propagation learning (Doya and Yoshizawa, 1989). Each CPG was composed of two neurons ($C = 2$) with time constants $\tau = 1.0$ and output functions $g(\,) = \tanh(\,)$. Instead of following the two step procedure described in the previous section, the network dynamics (5) and the learning equations (3) and (6) were simulated concurrently with parameters $\alpha = 0.1$, $\eta_0 = 0.2$, and $\tau_a = 20.0$.

Figure 1 **a** shows the oscillation of two CPGs without coupling. Figures 1 **b** through **e** show the phase-locked waveforms after learning for 200 time units with different desired delay times.

## 4 ZERO-LEGGED LOCOMOTION

Next we applied the learning rule to the simplest locomotion system that involves a critical phase-lock between the state of the physical system and the motor command—a zero-legged locomotion system as shown in Figure 2 **a**.

The physical system is composed of a wheel and a weight that moves back and forth on a track fixed radially in the wheel. It rolls on the ground by changing its balance with the displacement of the weight. In order to move the wheel in a given direction, the weight must be moved at a specific phase with the rotation angle of the wheel. The motion equations are shown in Appendix.

First, a CPG network was trained to oscillate with a sinusoidal waveform of period $T = 1.0$ (Doya and Yoshizawa, 1989). The network consisted of one output and two hidden units ($C = 3$) with time constants $\tau_i = 0.2$ and output functions $g_i(\,) = \tanh(\,)$. Next, the output of the CPG was used to drive the weight with a force $f = f_{\max} g_1(x_1(t))$. The position $r$ and the velocity $\dot{r}$ of the weight and the rotation angle $(\cos\theta, \sin\theta)$ and the angular velocity of the wheel $\dot{\theta}$ were used as sensory feedback inputs $y_k(t)$ ($k = 1, \dots, 5$) after scaling to $[-1, 1]$.

In order to eliminate the effect of biases in $\mathbf{x}(t)$ and $\mathbf{y}(t)$, we used the following learning equations.

$$\frac{d}{dt}p_{ik} = \eta \left\{ (x_i(t) - \bar{x}_i(t)) - \sum_{j=1}^{S} p_{ij}(y_j(t) - \bar{y}_j(t)) \right\}(y_k(t) - \bar{y}_k(t)),$$

$$\tau_x \frac{d}{dt}\bar{x}_i(t) = -\bar{x}_i(t) + x_i(t), \qquad (7)$$

$$\tau_y \frac{d}{dt}\bar{y}_k(t) = -\bar{y}_k(t) + y_k(t).$$

The rotation speed of the wheel was employed as the performance index $z(t)$ after smoothing by the following equation.

$$\tau_s \frac{d}{dt}z(t) = -z(t) + \dot{\theta}(t).$$

The learning coefficient $\eta$ was modulated by equations (6). The time constants were $\tau_x = 4.0$, $\tau_y = 1.0$, $\tau_s = 1.0$, and $\tau_a = 4.0$. Each training run was started from a random configuration of the wheel and was finished after ten seconds.

**a**

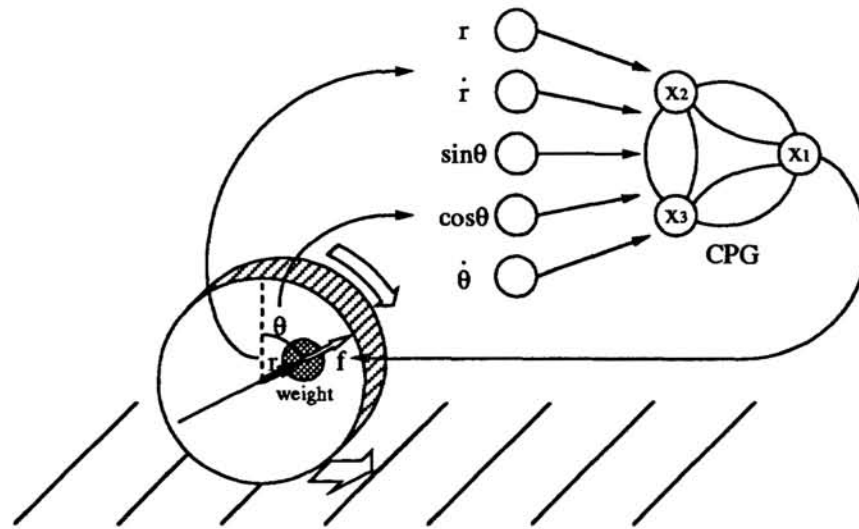

**b**

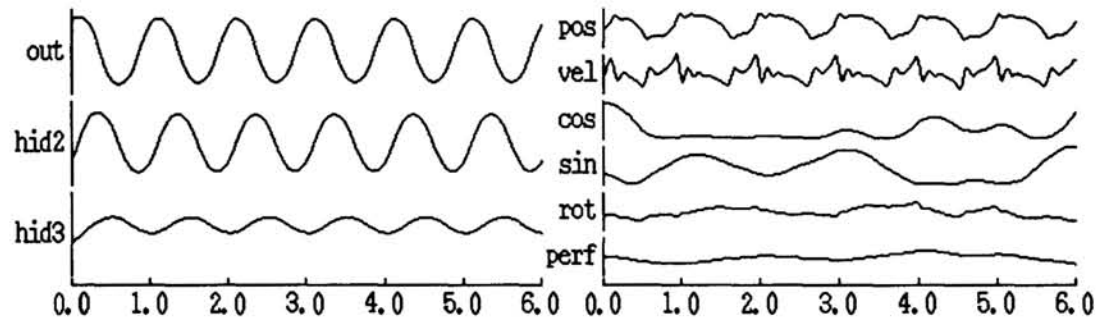

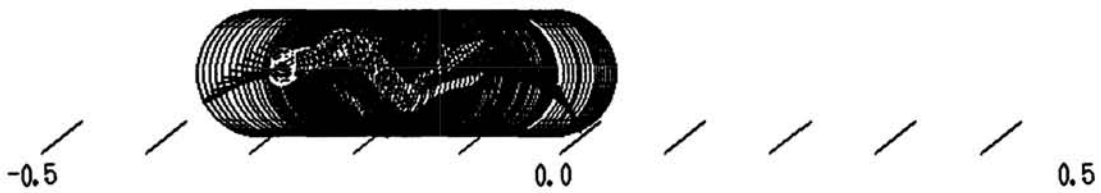

**c**

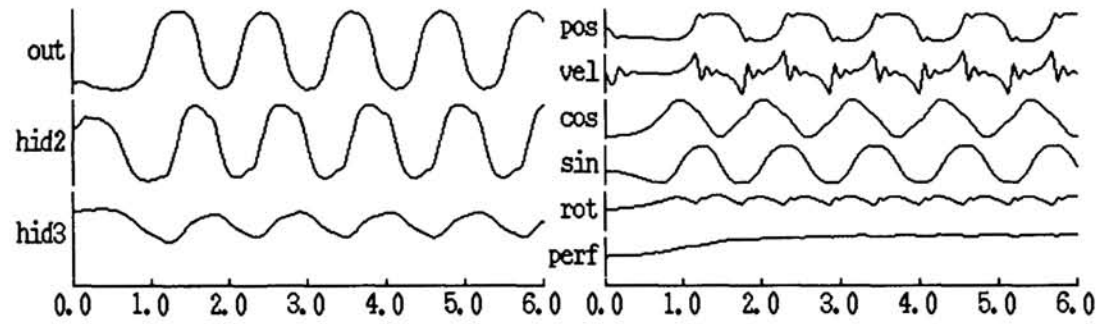

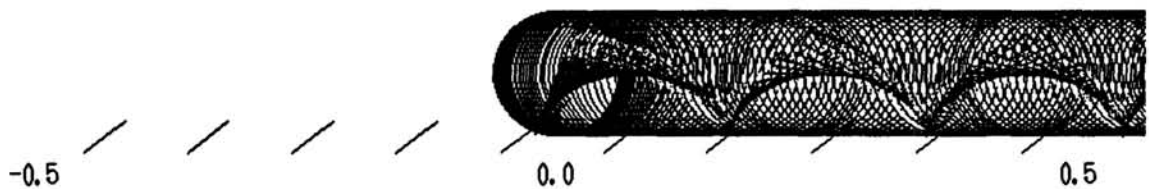

Figure 2: Learning of zero-legged locomotion.

Figure 2 b is an example of the motion of the wheel without sensory feedback. The rhythms of the CPG and the physical system were not entrained to each other and the wheel wandered left and right. Figure 2 c shows an example of the wheel motion after 40 runs of training with parameters $\eta_0 = 0.1$ and $\alpha = 0.2$. At first, the oscillation of the CPG was slowed down by the sensory inputs and then accelerated with the rotation of the wheel in the right direction.

We compared the patterns of sensory input connections made after learning with wheels of different sizes. Table 1 shows the connection weights to the output unit. The positive connection from $\sin\theta$ forces the weight to the right-hand side of the wheel and stabilize clockwise rotation. The negative connection from $\cos\theta$ with smaller radius fastens the rhythm of the CPG when the wheel rotates too fast and the weight is lifted up. The positive input from $r$ with larger radius makes the weight stickier to both ends of the track and slows down the rhythm of the CPG.

Table 1: Sensory input weights to the output unit ($p_{1k}; k = 1, \ldots, 5$).

| radius | $r$ | $\dot{r}$ | $\cos\theta$ | $\sin\theta$ | $\dot{\theta}$ |
|---|---|---|---|---|---|
| 2cm | 0.15 | -0.53 | -1.35 | 1.32 | 0.07 |
| 4cm | 0.28 | -0.55 | -1.09 | 1.22 | 0.01 |
| 6cm | 0.67 | -0.21 | -0.41 | 0.98 | 0.00 |
| 8cm | 0.70 | -0.33 | -0.40 | 0.92 | 0.03 |
| 10cm | 0.90 | -0.12 | -0.30 | 0.93 | -0.02 |

# 5  DISCUSSION

The architectures of CPGs in lower vertebrates and invertebrates are supposed to be determined by genetic information. Nevertheless, the way an animal utilizes the sensory inputs must be adaptive to the characteristics of the physical environments and the changing dimensions of its body parts.

Back-propagation through forward models of physical systems can also be applied to the learning of sensory feedback (Jordan and Jacobs, 1990). However, learning of nonlinear dynamics of locomotion systems is a difficult task; moreover, multi-layer back-propagation is not appropriate as a biological model of learning. The learning rule (7) is similar to the covariance learning rule (Sejnowski and Stanton, 1990), which is a biological model of long term potentiation of synapses.

**Acknowledgements**

The authors thank Allen Selverston, Peter Rowat, and those who gave comments to our poster at NIPS Conference. This work was partly supported by grants from the Ministry of Education, Culture, and Science of Japan.

## References

Barnes, W. J. P. & Gladden, M. H. (1985) *Feedback and Motor Control in Invertebrates and Vertebrates.* Beckenham, Britain: Croom Helm.

Doya, K. & Yoshizawa, S. (1989) Adaptive neural oscillator using continuous-time back-propagation learning. *Neural Networks*, 2, 375–386.

Grillner, S. & Matsushima, T. (1991) The neural network underlying locomotion in Lamprey—Synaptic and cellular mechanisms. *Neuron*, 7(July), 1-15.

Jordan, M. I. & Jacobs, R. A. (1990) Learning to control an unstable system with forward modeling. In Touretzky, D. S. (ed.), *Advances in Neural Information Processing Systems 2.* San Mateo, CA: Morgan Kaufmann.

Kawato, M. & Suzuki, R. (1980) Two coupled neural oscillators as a model of the circadian pacemaker. *Journal of Theoretical Biology*, 86, 547–575.

Pearlmutter, B. A. (1989) Learning state space trajectories in recurrent neural networks. *Neural Computation*, 1, 263–269.

Sejnowski, T. J. & Stanton, P. K. (1990) Covariance storage in the Hippocampus. In Zornetzer, S. F. et al. (eds.), *An Introduction to Neural and Electronic Networks*, 365–377. San Diego, CA: Academic Press.

Siller, K. T., Skorupski, P., Elson, R. C., & Bush, M. H. (1986) Two identified afferent neurones entrain a central locomotor rhythm generator. *Nature*, 323, 440–443.

Widrow, B. & Stearns, S. D. (1985) *Adaptive Signal Processing.* Englewood Cliffs, NJ: Prentice Hall.

Williams, R. J. & Zipser, D. (1989) A learning algorithm for continually running fully recurrent neural networks. *Neural Computation*, 1, 270–280.

## Appendix

The dynamics of the zero-legged locomotion system:

$$m\ddot{r} = f_0(1 + \frac{mR^2 \sin^2 \theta}{I_0}) - mg_c(\cos \theta + \frac{mR\sin^2 \theta(r + R\cos \theta)}{I_0})$$
$$+ mR\sin \theta \frac{\nu + 2m\dot{r}(r + R\cos \theta)}{I_0}\dot{\theta} + mr\dot{\theta}^2,$$
$$I_0\ddot{\theta} = -f_0 R\sin \theta + mg_c \sin \theta(r + R\cos \theta) - (\nu + 2m\dot{r}(r + R\cos \theta))\dot{\theta},$$
$$f_0 = f_{\max} g(x_1(t)) - \sigma r^3 - \mu \dot{r},$$
$$I_0 = I + MR^2 + m(r + R\cos \theta)^2.$$

Parameters: the masses of the weight $m = 0.2$[kg] and the wheel $M = 0.8$[kg]; the radius of the wheel $R = 0.02 through 0.1$[m]; the inertial moment of the wheel $I = \frac{1}{2}MR^2$; the maximum force to the weight $f_{\max} = 5$[N]; the stiffness of the limiter of the weight $\sigma = 20/R^3$ [N/m³]; the damping coefficients of the weight motion $\mu = 0.2/R$ [N/(m/s)] and the wheel rotation $\nu = 0.05(M+m)R$ [N/(rad/s)].